# Lg DEPTH ESTIMATION AND RIPPLE FIRE CHARACTERIZATION USING ARTIFICIAL NEURAL NETWORKS

**John L. Perry and Douglas R. Baumgardt**
ENSCO, Inc.
Signal Analysis and Systems Division
5400 Port Royal Road
Springfield, Virginia 22151
(703) 321-9000, perry@dewey.css.gov

## Abstract

This study has demonstrated how artificial neural networks (ANNs) can be used to characterize seismic sources using high-frequency regional seismic data. We have taken the novel approach of using ANNs as a research tool for obtaining seismic source information, specifically depth of focus for earthquakes and ripple-fire characteristics for economic blasts, rather than as just a feature classifier between earthquake and explosion populations. Overall, we have found that ANNs have potential applications to seismic event characterization and identification, beyond just as a feature classifier. In future studies, these techniques should be applied to actual data of regional seismic events recorded at the new regional seismic arrays. The results of this study indicates that an ANN should be evaluated as part of an operational seismic event identification system.

# 1 INTRODUCTION

## 1.1 NEURAL NETWORKS FOR SEISMIC SOURCE ANALYSIS

In this study, we have explored the application of artificial neural networks (ANNs) for the characterization of seismic sources for the purpose of distinguishing between explosions and earthquakes. ANNs have usually been used as pattern matching algorithms, and recent studies have applied ANNs to standard classification between classes of earthquakes and explosions using waveform features (Dowla, et al, 1989), (Dysart and Pulli, 1990). However, in considering the current state-of-the-art in seismic event identification, we believe the most challenging problem is not to develop a superior classification method, but rather, to have a better understanding of the physics of seismic source and regional signal propagation.

Our approach to the problem has been to use ANN technology as a research tool for obtaining a better understanding of the phenomenology behind regional discrimination, with emphasis on high-frequency regional array data, as well as using ANNs as a pattern classifier. We have explored two applications of ANNs to seismic source characterization: (1) the use of ANNs for depth characterization and (2) the recognition of ripple-firing effects in economic explosions.

In the first study, we explored the possible use of the Lg cross-coherence matrix, measured at a regional array, as a "hidden discriminant" for event depth of focus. In the second study, we experimented with applying ANNs to the recognition of ripple-fire effects in the spectra of regional phases. Moreover, we also investigated how a small (around 5 Kt yield) possibly decoupled nuclear explosion, detonated as part of a ripple-fire sequence, would affect the spectral modulations observed at regional distances and how these effects could be identified by the ANN.

## 1.2    ANN DESCRIPTION

*MLP Architecture:*    The ANN that we used was a multilayer perceptron (MLP) architecture with a backpropagation training algorithm (Rumelhart, et al, 1986). The input layer is fully connected to the hidden layer, which is fully connected to the output layer. There are no connections within an individual layer. Each node communicates with another node through a weighted connection. Associated with each connection is a weight connecting input node to hidden node, and a weight connecting hidden node to output node. The output of "activation level" of a particular node is defined as the linear weighted sum of all its inputs. For an MLP, a sigmoidal transformation is applied to this weighted sum. Two layers of our network have activation levels.

*MLP Training:* The MLP uses a backpropagation training algorithm which employs an iterating process where an output error signal is propagated back through the network and used to modify weight values. Training involves presenting sweeps of input patterns to the network and backpropagating the error until it is minimized. It is the weight values that represent a trained network and which can be used in the recognition/classification phase.

*MLP Recognition:* Recognition, on the other hand, involves presenting a pattern to a trained network and propagating node activation levels uni-directionally from the input layer, through the hidden layer(s), to the output layer, and then selecting the class corresponding to the highest output (activation) signal.

## 2  Lg DEPTH ESTIMATION

In theory, the Lg phase, which is often the largest regional phase on the seismogram, should provide depth information because Lg results from the superposition of numerous normal modes in the crust, whose excitation is highly depth dependent. Some studies have shown that Lg amplitudes do depend on depth (Der and Baumgardt, 1989). However, the precise dependency of Lg amplitude on depth has been hard to establish because other effects in the crustal model, such as anelastic attenuation, can also affect the Lg wave amplitude.

In this study, we have considered if the Lg coherence, measured across a regional array, might show depth dependency. This idea is based on the fact that all the normal modes which comprise Lg propagate at different phase velocities. For multilayered media, the normal modes will have frequency-dependent phase velocities because of dispersion. Our method for studying this dependency is a neural network implementation of a technique, called *matched field processing*, which has been used in underwater acoustics for source water-depth estimation (Bucker, 1976), (Baggeroer, et al, 1988). This method consists of computing the spectral matrix of an emitted signal, in our case, Lg, and comparing it against the same spectral matrix for master events at different depths. In the past, various optimal methods have been developed for the matching process. In our study, we have investigated using a neural network to accomplish the matching.

## 2.1 SPECTRAL MATRIX CALCULATION AND MATCHED FIELD PROCESSING

The following is a description of how the spectral matrix is computed. First, the synthetic seismograms for each of the nine elements of the hypothetical array are Fourier transformed in some time window. If $S_i(\omega)$ is the Fourier transform of a time window for the $i$ the channel, then, the spectral matrix is written as, $H_{ij}(\omega)=S_i(\omega)S_j^*(\omega)$, where $S_i(\omega)=A_i e^{-j[\omega t+\phi_i(\omega)]}$, the index $j$ is the complex number, $\phi_i$ is the phase angle, and the * represents complex transpose. The elements, $a_{ik}$ of the spectral matrix can be written as $a_{ik}(\omega)=A_iA_k e^{-j[\phi_i-\phi_k]}$ where the exponential phase shift term is

$$\Phi_i(\omega) - \Phi_k(\omega) = -\frac{\omega}{c_n(\omega)}(x_i-x_k) = -\omega\tau^n_{ik}(\omega)$$

. $c_n(\omega)$ represents the phase velocity for mode $n$, which is a function of frequency because of dispersion, $x_i - x_k$ is the spatial separation of the $i$ th and $k$ th channels of the array, and $\tau^n_{ik}(\omega)$ is the time shift of mode $n$ at frequency $\omega$ across the two channels. The product of the synthetic eigenfunctions, $A_i$, and thus, the spectral matrix terms, are functions of source depth and model parameters.

The spectral matrix, $H_{ij}(\omega)$, can be computed for an entire synthetic waveform or for a window on a part of the waveform. The elements of the spectral matrix can be normalized by inter- or intra- window normalization so that its values range from 0 to 1.

## 2.2 ANN - MATCHED FIELD DEPTH ESTIMATION

Two different depth studies were performed during this effort. The first study evaluated using the ANN to classify deep (greater than 4 kilometers) vs. shallow (less than 4 kilometers) seismic events. The number of input nodes equaled the number of points in the spectral matrix which was 1620 (36 data points x 45 spectral elements after smoothing). The number of output nodes was dependent on the type of classification we wanted to perform. For the shallow-deep discrimination, we only required two output nodes, one for each class. Training the ANN involved compiling a training (exemplar) set of spectral matrices for various shallow and deep events and then presenting the training set to the ANN.

In the second study, we investigated if the ANN could be used to classify seismic events at different depths. Again, we used five windows on the Lg phase and implemented the

interwindow and intrawindow normalization procedure. The second network was trained with a seven-element depth vector, whose elements represent the depths of 1, 3, 6, 9, 12, 16, and 20 kilometers.

# 3   RIPPLE-FIRE CHARACTERIZATION

In this study, we wanted to determine if spectral modulations could be recognized by the neural network and if they could be attached to concepts relating to the source parameters of ripple-fired events. Previous studies have demonstrated how such patterns could be found by looking for time-independent spectral modulations (Baumgardt and Ziegler, 1989), (Hedlin, et al, 1990). In this study, we assumed such a pattern has been found and that the input to the ANN is one of the time-independent spectra. An additional issue we considered was whether it would be possible to hide a nuclear explosion in the ripple-fire pattern, and whether or not such a pattern might be recognizable by an ANN. In this study, as in the previous depth characterization study, we relied entirely on simulated data for training and test.

## 3.1  ANN - RIPPLE FIRE CHARACTERIZATION

We performed two ripple-fired studies which were designed to extract different parameters from the ripple-fired events. The two studies characterized the following parameters: 1) time delay (Experiment A), and 2) normal vs. anomalous (Experiment B). The purpose of the time delay study was to estimate the time delay between explosions irrespective of the number of explosions. The goal of the second study was to determine if the ANN could extract a "normal" ripple-fired explosion from a simulated nuclear explosion buried in a ripple-fired event.

The input nodes to all three networks consisted of the elements of the seismic spectra which had 256 data points, covering the frequency range of 0 to 20 Hz. All weights were initialized to random values in the range of [-0.5, 0.5] and all networks had their momentum term set to 0.9. The number of hidden units, learning rate, and number of output nodes varied between each experiment.

In Experiment A, the number of hidden units was 24, and we used a learning rate of 0.2. We used seven output nodes to represent seven different classes with time delays of 37.5, 62.5, 87.5, 112.5, 137.5, 162.5, and 187.5 ms. These delay times are the centers of the following delay time bins: 25-50 ms, 50-75 ms, 75-100 ms, 100-125 ms, 125-150 ms, 150-175 ms, and 175-200 ms. We used five examples of each class for training derived by varying the time delay by ±5 msec. Training involved presenting the five exemplars for each class to the ANN until the squared error approached zero, as we did in the depth discrimination study. The ANN was trained to return a high activation level in the bin closest to the delay time of the ripple-fire.

In Experiment B, we wanted to determine if the ANN could discern between normal and anomalous ripple-fire patterns. The ANN was trained with 50 hidden units and a learning rate of 0.1. There were 36 exemplars of each class, which resulted from all combinations of six time delays of 5, 6, 7, 8, 9, and 10 ms between individual shots and six time delays of 5, 6, 7, 8, 9, and 10 ms between rows of shots. Each time delay was also varied ±1.0 ms to simulate the effect of errors in the blasting delays.

Two output nodes were defined which represent *anomalous* or *normal* ripple-fire.  The *normal* ripple-fire class represented all the simulations done for the triangular pattern.  We assumed each shot had a yield of 1000 kg.  The *anomalous* class were all the simulations for when the last row of 10 shots was replaced with a single large explosion of 10,000 kg.  The ANN was then trained to produce a high activation level for either of these classes depending on which kind of event it was presented.  The effect of the single large explosion signal was to wash out the scalloping pattern produced by the ripple-fired explosions.  We trained the ANN with normal ripple-fired patterns, with no embedded nuclear explosions, and anomalous patterns, with an embedded nuclear explosion.

# 4  RESULTS

## 4.1  RESULTS OF DEPTH STUDY

In our first study, we wanted to determine if the network could learn the simple concepts of *shallow* and *deep* from Lg synthetics when presented with only a small number of exemplar patterns.  We presented the ANN with four depths, 1, 2, 12, and 20 km, and trained the network to recognize the first two as shallow and the second two as deep.  We then presented the network with the rest of the synthetics, including synthetics for depths the ANN had not seen in training.

The results of the shallow-deep discrimination study are shown in Table 1.  The table shows the results for both the interwindow and intrawindow normalization procedures.  The test set used to generate these results were also synthetically generated events that were either less than 4 km (shallow) or greater than 4 km (deep).  Our criteria for a correct match was if the correct output node had an activation level that was 0.4 or more above the other output node's activation.  This is a very conservative threshold criteria, which is evident from the number of undecided values.  However, the results do indicate that the percent of incorrect classifications was only 5.0% for the intrawindow case and 8.3% for the interwindow case.  The percent of correct classification (PCC) for the intrawindow case was 50% and the PCC for the interwindow case was 58.3%.  The network appeared to be well trained, relative to their squared error values for this study.  Using a less conservative correct match criteria, where the correct output node only had to be larger than the other output node's activation, the PCC was 88.3% for the intrawindow case and 93.3% for the interwindow case.

| Depth (km.) | Intra-Window / Inter-Window | | | | | |
| --- | --- | --- | --- | --- | --- | --- |
| | correct | | incorrect | | undecided | |
| | 0.4 criterion | greatest activation | 0.4 criterion | greatest activation | 0.4 criterion | greatest activation |
| 3 | 2/3 | 5/5 | 0/0 | 0/0 | 3/2 | 0/0 |
| 4 | 1/2 | 3/3 | 0/2 | 0/2 | 4/1 | 2/0 |
| 5 | 2/3 | 4/5 | 0/0 | 0/0 | 3/2 | 1/0 |
| 6 | 4/3 | 5/5 | 0/1 | 0/0 | 1/1 | 0/0 |
| 7 | 2/4 | 4/5 | 1/0 | 1/0 | 2/1 | 0/0 |
| 8 | 3/2 | 5/5 | 0/0 | 0/0 | 2/3 | 0/0 |
| 9 | 3/5 | 4/5 | 1/0 | 0/0 | 1/0 | 1/0 |
| 10 | 3/3 | 5/5 | 0/0 | 0/0 | 2/2 | 0/0 |
| 11 | 1/2 | 4/3 | 1/2 | 1/2 | 3/1 | 0/0 |
| 13 | 2/2 | 5/5 | 0/0 | 0/0 | 3/3 | 0/0 |
| 14 | 4/4 | 5/5 | 0/0 | 0/0 | 1/1 | 0/0 |
| 15 | 3/2 | 4/5 | 0/0 | 0/0 | 2/3 | 1/0 |
| Total | 30/35 | 53/56 | 3/5 | 2/4 | 27/20 | 5/0 |

Table 1:  Results of ANN for Shallow-Deep Discrimination.

## 4.2 RESULTS OF THE RIPPLE-FIRED STUDY

*Linear Shot Patterns (Experiment A)*

Table 2 summarizes all the results for the time-delay ripple-fired classification study performed during Experiment A. The table shows both two-shot training and a two- and three-shot training cases. The test set for both cases were spectra that had time delays that were in a ±5 ms range of the target time delay pattern. We set two criteria for PCC for the two-shot case. The first was that the activation level for the correct output node be larger than the activation levels of the other output nodes. This produced a PCC of 77.7%, with a 22.2% error rate and no undecided responses. All of the errors resulted from attempting to use the ANN to learn time delays from a three-shot pattern where the network was only trained on two-shot events. The second criterion was more conservative and required that the activation level of the correct output node be ≥ 0.5 than the other output nodes. This gave a PCC = 68.2%, an error percentage of 4.5%, although the number of undecided responses increased to 27.2%. Again, all the errors resulted from expecting the ANN to generalize to three-shot events from only being trained with two-shot patterns. Finally, the results for the two- and three-shot training case were much more impressive. Using both threshold criteria, the ANN achieved a PCC of 100%.

| Test Set * | Threshold Criteria 0.0 / 0.5 | | |
|---|---|---|---|
| | correct | incorrect | undecided |
| Case A | 7 / 6 | 0 / 0 | 0 / 1 |
| Case B | 8 / 7 | 0 / 0 | 0 / 1 |
| 3 shots | 3 / 2 | 4 / 1 | 0 / 4 |
| Total | 13 / 15 | 4 / 1 | 0 / 6 |

\* (Trained with a 2-shot pattern)

| Test Set * | Threshold Criteria 0.0 / 0.5 | | |
|---|---|---|---|
| | correct | incorrect | undecided |
| Case A | 7 / 7 | 0 / 0 | 0 / 0 |
| Case B | 8 / 8 | 0 / 0 | 0 / 0 |
| 3 shots | 7 / 7 | 0 / 0 | 0 / 0 |
| Total | 22 / 22 | 0 / 0 | 0 / 0 |

\* (Trained with a 2- and 3-shot pattern)

Table 2: Results of ANN for Time-Delay Ripple-Fired Discrimination

*Triangular Shot Patterns - Normal Versus Anomalous (Experiment B)*

Table 3 depicts the results of Experiment B for the normal vs. anomalous study. The threshold criteria for the target output node compared to the other output nodes was 0.4. Again, the test set consisted of time delays that were within ±5 ms of the target time delay pattern. The PCC was 69.4%, the error percentage was 2.7%, and the percentage of undecided responses was 27.7%. As evident from the table, the majority of undecided responses were generated from attempting to classify the anomalous event.

| Test Set | Threshold Criteria 0.4 | | |
|---|---|---|---|
| | correct | incorrect | undecided |
| Normal | 31 | 1 | 4 |
| Anomalous | 19 | 1 | 16 |
| Total | 50 | 2 | 20 |

Table 3: Results of ANN for Normal vs. Anomalous Ripple-Fired Discrimination

## 5    CONCLUSIONS

This study has shown that ANNs can be used to characterize seismic waveform patterns for the purpose of characterizing depth of focus, from Lg spectral matrices, and for recognizing ripple-fire patterns from spectral modulations. However, we were only able to analyze the results for simulated input data. In future studies, we intend to use real data as input.

We have demonstrated that events can be classed as shallow or deep on the basis of the Lg spectral matrix and that the ANN provided a convenient and robust methodology for matching spectral matrices. The fact that we obtained nearly the same recognition performance for interwindow and intrawindow normalizations shows that the Lg spectral matrix does in fact contain significant information about the depth of focus of a seismic event, at least for theoretically derived synthetic cases.

The results for the ripple-fire recognition study were very encouraging. We found that neural networks could easily be trained to recognize many different ripple-fire patterns. For a given blasting region, a neural network could be trained to recognize the usual, routine ripple-fire patterns generally used in the region. We have shown that it should be possible to identify unusual or anomalous ripple-fire patterns due to attempts to include a large decoupled nuclear explosion in with an ordinary ripple-fire sequence.

## References

Baggeroer, A.M., W.A. Kuperman, and H. Schmidt (1988). Matched field processing: source localization in correlated noise as optimum parameter estimation, *J. Acoust. Soc. Am.*, **83**, 571-587.

Baumgardt, D.R. and K.A. Ziegler (1989). Automatic recognition of economic and underwater blasts using regional array data. *Unpublished report to Science Applications Incorporated, 11-880085-51.*

Bucker, H.P. (1976). Use of calculated sound fields and matched-field detection to locate sound sources in shallow water, *J. Acoust. Soc. Am.*, **59**, 368-373.

Der, Z.A. and D.R. Baumgardt (1989). Effect of source depth on the Lg phase, DARPA/AFTAC Research Review, November 1989.

Dowla, F.U., S.R. Taylor, and R.W. Anderson (1989). Seismic discrimination with artificial neural networks: preliminary results with regional spectral data, UCRL-102310, Lawrence Livermore National Laboratory, Livermore, CA.

Dysart, P.S. and J.J. Pulli (1990). Regional seismic event classification at the NORESS array; seismological measurements and the use of trained neural networks, abstract in *Program, Symposium on Regional Seismic Arrays and Nuclear Test Ban Verification*, Oslo, Norway, 14-17 February 1990.

Hedlin, M.A.H., J.B. Minster, J.A. Orcutt (1990). An automatic means to discriminate between earthquakes and quarry blasts, submitted to *Bull. Seism. Soc. Am.*

Rumelhart, D.E., Hinton, G.E., Williams, R.J. (1986). Learning internal representations by error propagation", in *Parallel Distributed Processing*, 1, MIT Press, Cambridge, MA.
